# A Constraint Generation Approach to Learning Stable Linear Dynamical Systems

**Sajid M. Siddiqi**
Robotics Institute
Carnegie-Mellon University
Pittsburgh, PA 15213
siddiqi@cs.cmu.edu

**Byron Boots**
Computer Science Department
Carnegie-Mellon University
Pittsburgh, PA 15213
beb@cs.cmu.edu

**Geoffrey J. Gordon**
Machine Learning Department
Carnegie-Mellon University
Pittsburgh, PA 15213
ggordon@cs.cmu.edu

## Abstract

Stability is a desirable characteristic for linear dynamical systems, but it is often ignored by algorithms that learn these systems from data. We propose a novel method for learning stable linear dynamical systems: we formulate an approximation of the problem as a convex program, start with a solution to a relaxed version of the program, and incrementally add constraints to improve stability. Rather than continuing to generate constraints until we reach a feasible solution, we test stability at each step; because the convex program is only an approximation of the desired problem, this early stopping rule can yield a higher-quality solution. We apply our algorithm to the task of learning dynamic textures from image sequences as well as to modeling biosurveillance drug-sales data. The constraint generation approach leads to noticeable improvement in the quality of simulated sequences. We compare our method to those of Lacy and Bernstein [1, 2], with positive results in terms of accuracy, quality of simulated sequences, and efficiency.

## 1 Introduction

Many problems in machine learning involve sequences of real-valued multivariate observations. To model the statistical properties of such data, it is often sensible to assume each observation to be correlated to the value of an underlying latent variable, or *state*, that is evolving over the course of the sequence. In the case where the state is real-valued and the noise terms are assumed to be Gaussian, the resulting model is called a *linear dynamical system* (LDS), also known as a Kalman Filter [3]. LDSs are an important tool for modeling time series in engineering, controls and economics as well as the physical and social sciences.

Let $\{\lambda_i(M)\}_{i=1}^n$ denote the eigenvalues of an $n \times n$ matrix $M$ in decreasing order of magnitude, $\{\nu_i(M)\}_{i=1}^n$ the corresponding unit-length eigenvectors, and define its *spectral radius* $\rho(M) \equiv |\lambda_1(M)|$. An LDS with dynamics matrix $A$ is *stable* if all of $A$'s eigenvalues have magnitude at most 1, i.e., $\rho(A) \leq 1$. Standard algorithms for learning LDS parameters do not enforce this stability criterion, learning locally optimal values for LDS parameters by gradient descent [4], Expectation Maximization (EM) [5] or least squares on a state sequence estimate obtained by subspace identification methods, as described in Section 3.1. However, when learning from finite data samples, the least squares solution may be unstable even if the system is stable [6]. The drawback of ignoring stability is most apparent when simulating long sequences from the system in order to generate representative data or infer stretches of missing values.

We propose a convex optimization algorithm for learning the dynamics matrix while guaranteeing stability. An estimate of the underlying state sequence is first obtained using subspace identification. We then formulate the least-squares problem for the dynamics matrix as a quadratic program (QP) [7], initially without constraints. When this QP is solved, the estimate $\hat{A}$ obtained may be unstable. However, any unstable solution allows us to derive a linear constraint which we then add

to our original QP and re-solve. The above two steps are iterated until we reach a stable solution, which is then refined by a simple interpolation to obtain the best possible stable estimate.

Our method can be viewed as *constraint generation* for an underlying convex program with a feasible set of all matrices with singular values at most 1, similar to work in control systems [1]. However, we terminate *before* reaching feasibility in the convex program, by checking for matrix stability after each new constraint. This makes our algorithm less conservative than previous methods for enforcing stability since it chooses the best of a larger set of stable dynamics matrices. The difference in the resulting stable systems is noticeable when simulating data. The constraint generation approach also achieves much greater efficiency than previous methods in our experiments.

One application of LDSs in computer vision is learning *dynamic textures* from video data [8]. An advantage of learning dynamic textures is the ability to play back a realistic-looking generated sequence of any desired duration. In practice, however, videos synthesized from dynamic texture models can quickly degenerate because of instability in the underlying LDS. In contrast, sequences generated from dynamic textures learned by our method remain "sane" even after arbitrarily long durations. We also apply our algorithm to learning baseline dynamic models of over-the-counter (OTC) drug sales for biosurveillance, and sunspot numbers from the UCR archive [9]. Comparison to the best alternative methods [1, 2] on these problems yields positive results.

## 2  Related Work

Linear system identification is a well-studied subject [4]. Within this area, *subspace identification methods* [10] have been very successful. These techniques first estimate the model dimensionality and the underlying state sequence, and then derive parameter estimates using least squares. Within subspace methods, techniques have been developed to enforce stability by augmenting the extended observability matrix with zeros [6] or adding a regularization term to the least squares objective [11].

All previous methods were outperformed by Lacy and Bernstein [1], henceforth referred to as LB-1. They formulate the problem as a semidefinite program (SDP) whose objective minimizes the state sequence reconstruction error, and whose constraint bounds the largest singular value by 1. This convex constraint is obtained by rewriting the nonlinear matrix inequality $I_n - AA^T \succeq 0$ as a linear matrix inequality [12], where $I_n$ is the $n \times n$ identity matrix. Here, $\succ 0$ ($\succeq 0$) denotes positive (semi-) definiteness. The existence of this constraint also proves the convexity of the $\sigma_1 \leq 1$ region.

A follow-up to this work by the same authors [2], which we will call LB-2, attempts to overcome the conservativeness of LB-1 by approximating the Lyapunov inequalities $P - APA^T \succ 0, P \succ 0$ with the inequalities $P - APA^T - \delta I_n \succeq 0, P - \delta I_n \succeq 0, \delta > 0$. These inequalities hold iff the spectral radius is less than 1. However, the approximation is achieved only at the cost of inducing a nonlinear distortion of the objective function by a problem-dependent reweighting matrix involving $P$, which is a variable to be optimized. In our experiments, this causes LB-2 to perform worse than LB-1 (for any $\delta$) in terms of the state sequence reconstruction error, even while obtaining solutions outside the feasible region of LB-1. Consequently, we focus on LB-1 in our conceptual and qualitative comparisons as it is the strongest baseline available. However, LB-2 is more scalable than LB-1, so quantitative results are presented for both.

To summarize the distinction between constraint generation, LB-1 and LB-2: it is hard to have both the right objective function (reconstruction error) and the right feasible region (the set of stable matrices). LB-1 optimizes the right objective but over the wrong feasible region (the set of matrices with $\sigma_1 \leq 1$). LB-2 has a feasible region close to the right one, but at the cost of distorting its objective function to an extent that it fares worse than LB-1 in nearly all cases. In contrast, our method optimizes the right objective over a less conservative feasible region than that of any previous algorithm with the right objective, and this combination is shown to work the best in practice.

## 3  Linear Dynamical Systems

The evolution of a linear dynamical system can be described by the following two equations:

$$x_{t+1} = Ax_t + w_t$$
$$y_t = Cx_t + v_t \tag{1}$$

Time is indexed by the discrete variable $t$. Here $x_t$ denotes the hidden states in $\mathbb{R}^n$, $y_t$ the observations in $\mathbb{R}^m$, and $w_t$ and $v_t$ are zero-mean normally distributed state and observation noise variables.

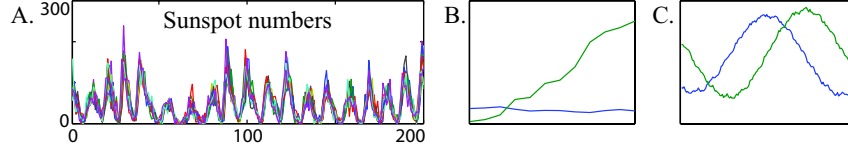

Figure 1: A. Sunspot data, sampled monthly for 200 years. Each curve is a month, the $x$-axis is over years. B. First two principal components of a 1-observation Hankel matrix. C. First two principal components of a 12-observation Hankel matrix, which better reflect temporal patterns in the data.

Assume some initial state $x_0$. The parameters of the system are the dynamics matrix $A \in \mathbb{R}^{n \times n}$, the observation model $C \in \mathbb{R}^{m \times n}$, and the noise covariance matrices $Q$ and $R$. Note that we are learning *uncontrolled* linear dynamical systems, though, as in previous work, control inputs can easily be incorporated into the objective function and convex program.

Linear dynamical systems can also be viewed as probabilistic graphical models. The standard LDS filtering and smoothing inference algorithms [3, 13] are instantiations of the junction tree algorithm for Bayesian Networks (see, for example, [14]).

We follow the subspace identification literature in estimating all parameters other than the dynamics matrix. A clear and concise exposition of the required techniques is presented in Soatto et al. [8], which we summarize below. We use subspace identification methods in our experiments for uniformity with previous work we are building on (in the control systems literature) and with work we are comparing to ([8] on the dynamic textures data).

### 3.1 Learning Model Parameters by Subspace Methods

Subspace methods calculate LDS parameters by first decomposing a matrix of observations to yield an estimate of the underlying state sequence. The most straightforward such technique is used here, which relies on the *singular value decomposition* (SVD) [15]. See [10] for variations.

Let $Y_{1:\tau} = [y_1 \ y_2 \ \dots \ y_\tau] \in \mathbb{R}^{m \times \tau}$ and $X_{1:\tau} = [x_1 \ x_2 \ \dots \ x_\tau] \in \mathbb{R}^{n \times \tau}$. $\mathcal{D}$ denotes the matrix of observations which is the input to SVD. One typical choice for $\mathcal{D}$ is $\mathcal{D} = Y_{1:\tau}$; we will discuss others below. SVD yields $\mathcal{D} \approx U\Sigma V^T$ where $U \in \mathbb{R}^{m \times n}$ and $V \in \mathbb{R}^{\tau \times n}$ have orthonormal columns $\{u_i\}$ and $\{v_i\}$, and $\Sigma = \text{diag}\{\sigma_1, \dots, \sigma_n\}$ contains the singular values. The model dimension $n$ is determined by keeping all singular values of $\mathcal{D}$ above a threshold. We obtain estimates of $C$ and $X$:

$$\hat{C} = U \qquad \qquad \hat{X} = \Sigma V^T \tag{2}$$

See [8] for an explanation of why these estimates satisfy certain *canonical model* assumptions. $\hat{X}$ is referred to as the *extended observability matrix* in the control systems literature; the $t^{th}$ column of $\hat{X}$ represents an estimate of the state of our LDS at time $t$. The least squares estimate of $A$ is:

$$\hat{A} = \arg\min_A J^2(A) = \arg\min_A \left\| AX_{0:\tau-1} - X_{1:\tau} \right\|_F^2 = X_{1:\tau}X_{0:\tau-1}^\dagger \tag{3}$$

where $\| \cdot \|_F$ denotes the Frobenius norm and $^\dagger$ denotes the Moore-Penrose inverse. Eq. (3) asks $\hat{A}$ to minimize the error in predicting the state at time $t+1$ from the state at time $t$. Given the above estimates $\hat{A}$ and $\hat{C}$, the covariance matrices $\hat{Q}$ and $\hat{R}$ can be estimated directly from residuals.

### 3.2 Designing the Observation Matrix

In the decomposition above, we chose each column of $\mathcal{D}$ to be the observation vector for a single time step. Suppose that instead we set $\mathcal{D}$ to be a matrix of the form

$$\mathcal{D} = \left[ \begin{array}{ccccc} y_1 & y_2 & y_3 & \cdots & y_\tau \\ \vdots & \vdots & \vdots & \ddots & \vdots \\ y_d & y_{d+1} & y_{d+2} & \cdots & y_{d+\tau-1} \end{array} \right]_{md \times \tau}$$

A matrix of this form, with each block of rows equal to the previous block but shifted by a constant number of columns, is called a *block Hankel* matrix [4]. We say "$d$-observation Hankel matrix of size $\tau$" to mean the data matrix $\mathcal{D} \in \mathbb{R}^{md \times \tau}$ with $d$ length-$m$ observation vectors per column. Stacking observations causes each state to incorporate more information about the future, since $\hat{x}_t$

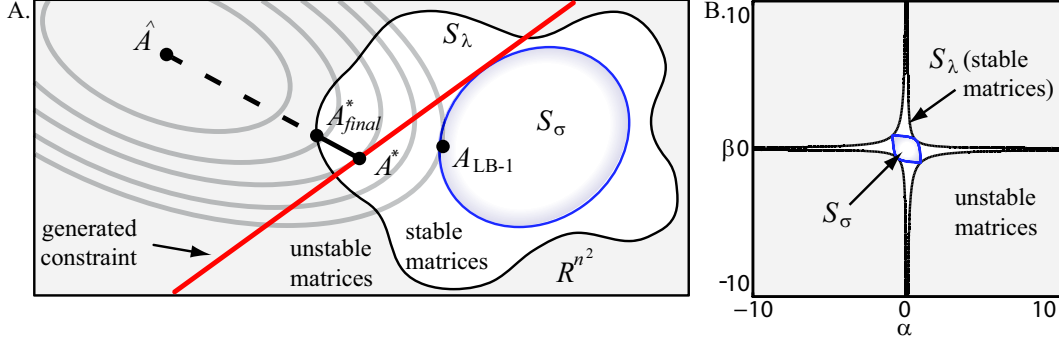

Figure 2: (A): Conceptual depiction of the space of $n \times n$ matrices. The region of stability ($S_\lambda$) is non-convex while the smaller region of matrices with $\sigma_1 \leq 1$ ($S_\sigma$) is convex. The elliptical contours indicate level sets of the quadratic objective function of the QP. $\hat{A}$ is the unconstrained least-squares solution to this objective. $A_{\text{LB-1}}$ is the solution found by LB-1 [1]. One iteration of constraint generation yields the constraint indicated by the line labeled 'generated constraint', and (in this case) leads to a stable solution $A^*$. The final step of our algorithm improves on this solution by interpolating $A^*$ with the previous solution (in this case, $\hat{A}$) to obtain $A^*_{final}$. (B): The actual stable and unstable regions for the space of $2 \times 2$ matrices $E_{\alpha,\beta} = [\, 0.3 \ \alpha \,; \beta \ 0.3 \,]$, with $\alpha, \beta \in [-10, 10]$. Constraint generation is able to learn a nearly optimal model from a noisy state sequence of length 7 simulated from $E_{0,10}$, with better state reconstruction error than either LB-1 or LB-2.

now represents coefficients reconstructing $y_t$ *as well as* other observations in the future. However the observation model estimate must now be $\hat{C} = U(\, : , 1\!:\!m)$, i.e., the submatrix consisting of the first $m$ columns of $U$, because $U(\, : , 1\!:\!m)\hat{x}_t = \hat{y}_t$ for any $t$, where $\hat{y}_t$ denotes a reconstructed observation. Having multiple observations per column in $\mathcal{D}$ is particularly helpful when the underlying dynamical system is known to have periodicity. For example, see Figure 1(A). See [12] for details.

## 4 The Algorithm

The estimation procedure in Section 3.1 does not enforce stability in $\hat{A}$. To account for stability, we first formulate the dynamics matrix learning problem as a quadratic program with a feasible set that includes the set of stable dynamics matrices. Then we demonstrate how instability in its solutions can be used to generate constraints that restrict this feasible set appropriately. As a final step, the solution is refined to be as close as possible to the least-squares estimate while remaining stable. The overall algorithm is illustrated in Figure 2(A). We now explain the algorithm in more detail.

### 4.1 Formulating the Objective

The least squares problem in Eq. (3) can be written as follows (see [12] for the derivation):

$$\hat{A} = \arg\min_A \left\| AX_{0:\tau-1} - X_{1:\tau} \right\|_F^2$$
$$= \arg\min_a \left\{ a^T Pa - 2\, q^T a + r \right\} \tag{4}$$

where $a \in \mathbb{R}^{n^2 \times 1}$, $q \in \mathbb{R}^{n^2 \times 1}$, $P \in \mathbb{R}^{n^2 \times n^2}$ and $r \in \mathbb{R}$ are defined as:

$$a = \text{vec}(A) = [A_{11}\ A_{21}\ A_{31}\ \cdots\ A_{nn}]^T \qquad P = I_n \otimes \left( X_{0:\tau-1} X_{0:\tau-1}^T \right)$$
$$q = \text{vec}(X_{0:\tau-1} X_{1:\tau}^T) \qquad\qquad r = \text{tr}\left( X_{1:\tau}^T X_{1:\tau} \right) \tag{5}$$

$I_n$ is the $n \times n$ identity matrix and $\otimes$ denotes the Kronecker product. Note that $P$ is a symmetric nonnegative-definite matrix. The objective function in (4) is a quadratic function of $a$.

### 4.2 Generating Constraints

The quadratic objective function above is equivalent to the least squares problem of Eq. (3). Its feasible set is the space of all $n \times n$ matrices, regardless of their stability. When its solution yields an unstable matrix, the spectral radius of $\hat{A}$ (i.e. $|\lambda_1(\hat{A})|$) is greater than 1. Ideally we would like to use $\hat{A}$ to calculate a convex constraint on the spectral radius. However, consider the class of $2 \times 2$ matrices [16]: $E_{\alpha,\beta} = [\, 0.3 \ \alpha \,; \beta \ 0.3 \,]$. The matrices $E_{10,0}$ and $E_{0,10}$ are stable with $\lambda_1 = 0.3$, but

their convex combination $\gamma E_{10,0} + (1 - \gamma)E_{0,10}$ is unstable for (e.g.) $\gamma = 0.5$ (Figure 2(B)). This shows that the set of stable matrices is non-convex for $n = 2$, and in fact this is true for all $n > 1$. We turn instead to the largest *singular value*, which is a closely related quantity since

$$\sigma_{min}(\hat{A}) \leq |\lambda_i(\hat{A})| \leq \sigma_{max}(\hat{A}) \quad \forall i = 1, \ldots, n \qquad [15]$$

Therefore every unstable matrix has a singular value greater than one, but the converse is not necessarily true. Moreover, the set of matrices with $\sigma_1 \leq 1$ *is* convex. Figure 2(A) conceptually depicts the non-convex region of stability $S_\lambda$ and the convex region $S_\sigma$ with $\sigma_1 \leq 1$ in the space of all $n \times n$ matrices for some fixed $n$. The difference between $S_\sigma$ and $S_\lambda$ can be significant. Figure 2(B) depicts these regions for $E_{\alpha,\beta}$ with $\alpha, \beta \in [-10, 10]$. The stable matrices $E_{10,0}$ and $E_{0,10}$ reside at the edges of the figure. While results for this class of matrices vary, the constraint generation algorithm described below is able to learn a nearly optimal model from a noisy state sequence of $\tau = 7$ simulated from $E_{0,10}$, with better state reconstruction error than LB-1 and LB-2.

Let $\hat{A} = \tilde{U}\tilde{\Sigma}\tilde{V}^T$ by SVD, where $\tilde{U} = [\tilde{u}_i]_{i=1}^n$ and $\tilde{V} = [\tilde{v}_i]_{i=1}^n$ and $\tilde{\Sigma} = \text{diag}\{\tilde{\sigma}_1, \ldots, \tilde{\sigma}_n\}$. Then:

$$\hat{A} = \tilde{U}\tilde{\Sigma}\tilde{V}^T \Rightarrow \quad \tilde{\Sigma} = \tilde{U}^T\hat{A}\tilde{V} \Rightarrow \quad \tilde{\sigma}_1(\hat{A}) = \tilde{u}_1^T\hat{A}\tilde{v}_1 = \text{tr}(\tilde{u}_1^T\hat{A}\tilde{v}_1) \qquad (6)$$

Therefore, instability of $\hat{A}$ implies that:

$$\tilde{\sigma}_1 > 1 \Rightarrow \quad \text{tr}\left(\tilde{u}_1^T\hat{A}\tilde{v}_1\right) > 1 \Rightarrow \quad \text{tr}\left(\tilde{v}_1\tilde{u}_1^T\hat{A}\right) > 1 \Rightarrow \quad g^T\hat{a} > 1 \qquad (7)$$

Here $g = \text{vec}(\tilde{u}_1\tilde{v}_1^T)$. Since Eq. (7) arose from an unstable solution of Eq. (4), $g$ is a hyperplane separating $\hat{a}$ from the space of matrices with $\sigma_1 \leq 1$. We use the negation of Eq. (7) as a constraint:

$$g^T\hat{a} \leq 1 \qquad (8)$$

### 4.3 Computing the Solution

The overall quadratic program can be stated as:

$$\begin{aligned} \text{minimize} \quad & a^TPa - 2\,q^Ta + r \\ \text{subject to} \quad & Ga \leq h \end{aligned} \qquad (9)$$

with $a$, $P$, $q$ and $r$ as defined in Eqs. (5). $\{G, h\}$ define the set of constraints, and are initially empty. The QP is invoked repeatedly until the stable region, i.e. $S_\lambda$, is reached. At each iteration, we calculate a linear constraint of the form in Eq. (8), add the corresponding $g^T$ as a row in $G$, and augment $h$ with 1. Note that we will almost always stop *before* reaching the feasible region $S_\sigma$. Once a stable matrix is obtained, it is possible to refine this solution. We know that the last constraint caused our solution to cross the boundary of $S_\lambda$, so we interpolate between the last solution and the previous iteration's solution using binary search to look for a boundary of the stable region, in order to obtain a better objective value while remaining stable. An interpolation could be attempted between the least squares solution and any stable solution. However, the stable region can be highly complex, and there may be several folds and boundaries of the stable region in the interpolated area. In our experiments (not shown), interpolating from the LB-1 solution yielded worse results.

## 5 Experiments

For learning the dynamics matrix, we implemented[1] least squares, constraint generation (using `quadprog`), LB-1 [1] and LB-2 [2] (using `CVX` with `SeDuMi`) in Matlab on a 3.2 GHz Pentium with 2 GB RAM. Note that these algorithms give a different result from the basic least-squares system identification algorithm only in situations where the least-squares model is unstable. However, least-squares LDSs trained in scarce-data scenarios are unstable for almost any domain, and some domains lead to unstable models up to the limit of available data (e.g. the `steam` dynamic textures in Section 5.1). The goals of our experiments are to: (1) examine the state evolution and simulated observations of models learned using our method, and compare them to previous work; and (2) compare the algorithms in terms of reconstruction error and efficiency. The error metric used for the quantitative experiments when evaluating matrix $A^*$ is

$$e_x(A^*) = 100 \times \left(J^2(A^*) - J^2(\hat{A})\right)/J^2(\hat{A}) \qquad (10)$$

i.e. percent increase in squared reconstruction error compared to least squares, with $J(\cdot)$ as defined in Eq. (4). We apply these algorithms to learning dynamic textures from the vision domain (Section 5.1), as well as OTC drug sales counts and sunspot numbers (Section 5.2).

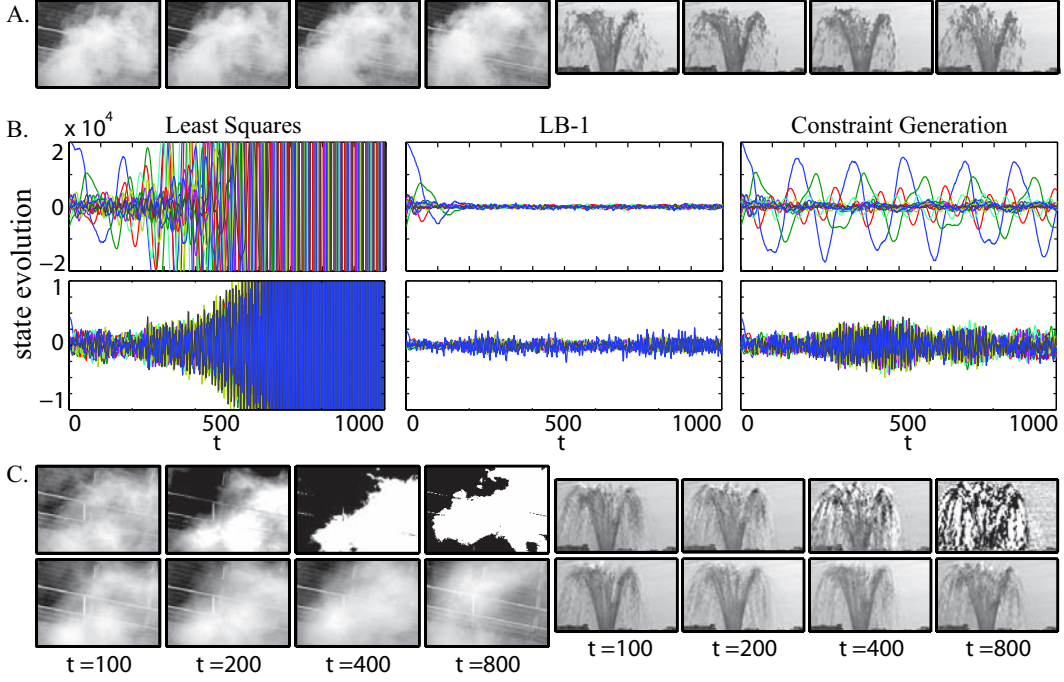

Figure 3: Dynamic textures. A. Samples from the original `steam` sequence and the `fountain` sequence. B. State evolution of synthesized sequences over 1000 frames (`steam` top, `fountain` bottom). The least squares solutions display instability as time progresses. The solutions obtained using LB-1 remain stable for the full 1000 frame image sequence. The constraint generation solutions, however, yield state sequences that are stable over the full 1000 frame image sequence without significant damping. C. Samples drawn from a least squares synthesized sequences (top), and samples drawn from a constraint generation synthesized sequence (bottom). Images for LB-1 are not shown. The constraint generation synthesized `steam` sequence is qualitatively better looking than the `steam` sequence generated by LB-1, although there is little qualitative difference between the two synthesized `fountain` sequences.

| | CG | LB-1 | LB-1* | LB-2 | CG | LB-1 | LB-1* | LB-2 |
|---|---|---|---|---|---|---|---|---|
| | `steam` ($n = 10$) | | | | `fountain` ($n = 10$) | | | |
| $|\lambda_1|$ | 1.000 | 0.993 | 0.993 | 1.000 | 0.999 | 0.987 | 0.987 | 0.997 |
| $\sigma_1$ | 1.036 | 1.000 | 1.000 | 1.034 | 1.051 | 1.000 | 1.000 | 1.054 |
| $e_x(\%)$ | **45.2** | 103.3 | 103.3 | 546.9 | **0.1** | 4.1 | 4.1 | 3.0 |
| time | **0.45** | 95.87 | 3.77 | 0.50 | **0.15** | 15.43 | 1.09 | 0.49 |
| | `steam` ($n = 20$) | | | | `fountain` ($n = 20$) | | | |
| $|\lambda_1|$ | 0.999 | — | 0.990 | 0.999 | 0.999 | — | 0.988 | 0.996 |
| $\sigma_1$ | 1.037 | — | 1.000 | 1.062 | 1.054 | — | 1.000 | 1.056 |
| $e_x(\%)$ | **58.4** | — | 154.7 | 294.8 | **1.2** | — | 5.0 | 22.3 |
| time | **2.37** | — | 1259.6 | 33.55 | **1.63** | — | 159.85 | 5.13 |
| | `steam` ($n = 40$) | | | | `fountain` ($n = 40$) | | | |
| $|\lambda_1|$ | 1.000 | — | 0.989 | 1.000 | 1.000 | — | 0.991 | 1.000 |
| $\sigma_1$ | 1.120 | — | 1.000 | 1.128 | 1.034 | — | 1.000 | 1.172 |
| $e_x(\%)$ | **20.24** | — | 282.7 | 768.5 | **3.3** | — | 4.8 | 21.5 |
| time | **5.85** | — | 79516.98 | 289.79 | **61.9** | — | 43457.77 | 239.53 |

Table 1: Quantitative results on the dynamic textures data for different numbers of states $n$. CG is our algorithm, LB-1and LB-2 are competing algorithms, and LB-1* is a simulation of LB-1 using our algorithm by generating constraints until we reach $S_\sigma$, since LB-1 failed for $n > 10$ due to memory limits. $e_x$ is percent difference in squared reconstruction error as defined in Eq. (10). Constraint generation, in all cases, has lower error and faster runtime.

## 5.1 Stable Dynamic Textures

Dynamic textures in vision can intuitively be described as models for sequences of images that exhibit some form of low-dimensional structure and recurrent (though not necessarily repeating) characteristics, e.g. fixed-background videos of rising smoke or flowing water. Treating each frame of a video as an observation vector of pixel values $y_t$, we learned dynamic texture models of two video sequences: the `steam` sequence, composed of $120 \times 170$ pixel images, and the `fountain` sequence, composed of $150 \times 90$ pixel images, both of which originated from the MIT temporal texture database (Figure 3(A)). We use parameters $\tau = 80$, $n = 15$, and $d = 10$. Note that the state sequence we learn has no *a priori* interpretation.

An LDS model of a dynamic texture may *synthesize* an "infinitely" long sequence of images by driving the model with zero mean Gaussian noise. Each of our two models uses an 80 frame training sequence to generate 1000 sequential images in this way. To better visualize the difference between image sequences generated by least-squares, LB-1, and constraint generation, the evolution of each method's state is plotted over the course of the synthesized sequences (Figure 3(B)). Sequences generated by the least squares models appear to be unstable, and this was in fact the case; both the `steam` and the `fountain` sequences resulted in unstable dynamics matrices. Conversely, the constrained subspace identification algorithms all produced well-behaved sequences of states and stable dynamics matrices (Table 1), although constraint generation demonstrates the fastest runtime, best scalability, and lowest error of any stability-enforcing approach.

A qualitative comparison of images generated by constraint generation and least squares (Figure 3(C)) indicates the effect of instability in synthesized sequences generated from dynamic texture models. While the unstable least-squares model demonstrates a dramatic increase in image contrast over time, the constraint generation model continues to generate qualitatively reasonable images. Qualitative comparisons between constraint generation and LB-1 indicate that constraint generation learns models that generate more natural-looking video sequences[2] than LB-1.

Table 1 demonstrates that constraint generation always has the lowest error as well as the fastest runtime. The running time of constraint generation depends on the number of constraints needed to reach a stable solution. Note that LB-1 is more efficient and scalable when simulated using constraint generation (by adding constraints until $S_\sigma$ is reached) than it is in its original SDP formulation.

## 5.2 Stable Baseline Models for Biosurveillance

We examine daily counts of OTC drug sales in pharmacies, obtained from the National Data Retail Monitor (NDRM) collection [17]. The counts are divided into 23 different categories and are tracked separately for each zipcode in the country. We focus on zipcodes from a particular American city. The data exhibits 7-day periodicity due to differential buying patterns during the week. We isolate a 60-day subsequence where the data dynamics remain relatively stationary, and attempt to learn LDS parameters to be able to simulate sequences of baseline values for use in detecting anomalies.

We perform two experiments on different aggregations of the OTC data, with parameter values $n = 7, d = 7$ and $\tau = 14$. Figure 4(A) plots 22 different drug categories aggregated over all zipcodes, and Figure 4(B) plots a single drug category (cough/cold) in 29 different zipcodes separately. In both cases, constraint generation is able to use very little training data to learn a stable model that captures the periodicity in the data, while the least squares model is unstable and its predictions diverge over time. LB-1 learns a model that is stable but overconstrained, and the simulated observations quickly drift from the correct magnitudes. We also tested the algorithms on the sunspots data (Figure 2(C)) with parameters $n = 7, d = 18$ and $\tau = 50$, with similar results. Quantitative results on both these domains exhibit similar trends as those in Table 1.

# 6 Discussion

We have introduced a novel method for learning stable linear dynamical systems. Our constraint generation algorithm is more powerful than previous methods in the sense of optimizing over a larger set of stable matrices with a suitable objective function. The constraint generation approach also has the benefit of being faster than previous methods in nearly all of our experiments. One possible extension is to modify the EM algorithm for LDSs to incorporate constraint generation into the M-step in order to learn stable systems that locally maximize the observed data likelihood. Stability could also be of advantage in planning applications.

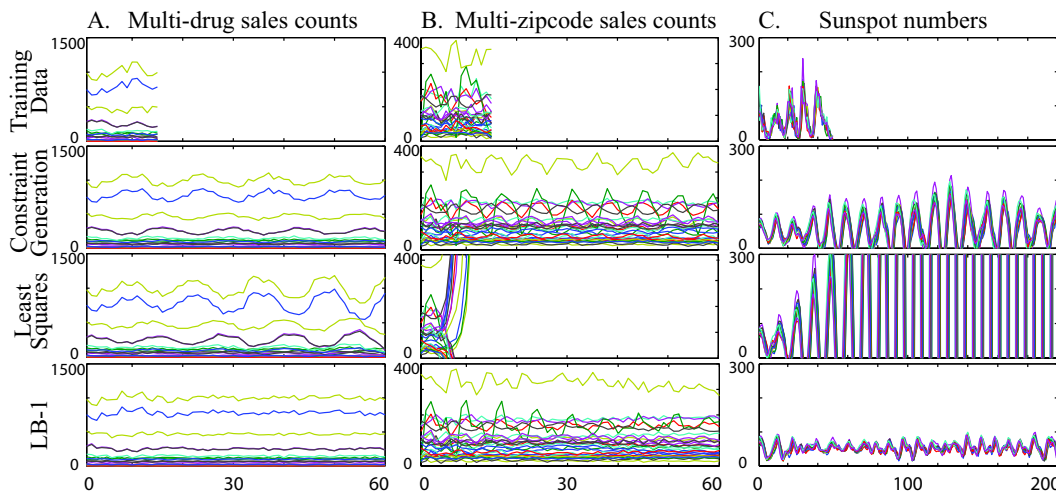

Figure 4: (A): 60 days of data for 22 drug categories aggregated over all zipcodes in the city. (B): 60 days of data for a single drug category (cough/cold) for all 29 zipcodes in the city. (C): Sunspot numbers for 200 years separately for each of the 12 months. The training data (top), simulated output from constraint generation, output from the unstable least squares model, and output from the over-damped LB-1 model (bottom).

## Acknowledgements

This paper is based on work supported by DARPA under the Computer Science Study Panel program (authors GJG and BEB), the NSF under Grant Nos. EEC-0540865 (author BEB) and IIS-0325581 (author SMS), and the CDC under award 8-R01-HK000020-02, "Efficient, scalable multisource surveillance algorithms for Biosense" (author SMS).

## Footnotes

[1] Source code is available at http://www.select.cs.cmu.edu/projects/stableLDS

[2]See videos at http://www.select.cs.cmu.edu/projects/stableLDS

## References

[1] Seth L. Lacy and Dennis S. Bernstein. Subspace identification with guaranteed stability using constrained optimization. In *Proc. American Control Conference*, 2002.

[2] Seth L. Lacy and Dennis S. Bernstein. Subspace identification with guaranteed stability using constrained optimization. *IEEE Transactions on Automatic Control*, 48(7):1259–1263, July 2003.

[3] R.E. Kalman. A new approach to linear filtering and prediction problems. *Trans. ASME–JBE*, 1960.

[4] L. Ljung. *System Identification: Theory for the user*. Prentice Hall, 2nd edition, 1999.

[5] Zoubin Ghahramani and Geoffrey E. Hinton. Parameter estimation for Linear Dynamical Systems. Technical Report CRG-TR-96-2, U. of Toronto, Department of Comp. Sci., 1996.

[6] N. L. C. Chui and J. M. Maciejowski. Realization of stable models with subspace methods. *Automatica*, 32(100):1587–1595, 1996.

[7] Stephen Boyd and Lieven Vandenberghe. *Convex Optimization*. Cambridge University Press, 2004.

[8] S. Soatto, G. Doretto, and Y. Wu. Dynamic Textures. *Intl. Conf. on Computer Vision*, 2001.

[9] E. Keogh and T. Folias. The UCR Time Series Data Mining Archive, 2002.

[10] P. Van Overschee and B. De Moor. *Subspace Identification for Linear Systems: Theory, Implementation, Applications*. Kluwer, 1996.

[11] T. Van Gestel, J. A. K. Suykens, P. Van Dooren, and B. De Moor. Identification of stable models in subspace identification by using regularization. *IEEE Transactions on Automatic Control*, 2001.

[12] Sajid M. Siddiqi, Byron Boots, and Geoffrey J. Gordon. A Constraint Generation Approach to Learning Stable Linear Dynamical Systems. Technical Report CMU-ML-08-101, CMU, 2008.

[13] H. Rauch. Solutions to the linear smoothing problem. In *IEEE Transactions on Automatic Control*, 1963.

[14] Kevin Murphy. *Dynamic Bayesian Networks*. PhD thesis, UC Berkeley, 2002.

[15] Roger Horn and Charles R. Johnson. *Matrix Analysis*. Cambridge University Press, 1985.

[16] Andrew Y. Ng and H. Jin Kim. Stable adaptive control with online learning. In *Proc. NIPS*, 2004.

[17] M. Wagner. A national retail data monitor for public health surveillance. *Morbidity and Mortality Weekly Report*, 53:40–42, 2004.
